# Spike timing-dependent plasticity as dynamic filter

**Joscha T. Schmiedt,**[*] **Christian Albers and Klaus Pawelzik**
Institute for Theoretical Physics
University of Bremen
Bremen, Germany
schmiedt@uni-bremen.de, {calbers, pawelzik}@neuro.uni-bremen.de

## Abstract

When stimulated with complex action potential sequences synapses exhibit spike timing-dependent plasticity (STDP) with modulated pre- and postsynaptic contributions to long-term synaptic modifications. In order to investigate the functional consequences of these contribution dynamics (CD) we propose a minimal model formulated in terms of differential equations. We find that our model reproduces data from to recent experimental studies with a small number of biophysically interpretable parameters. The model allows to investigate the susceptibility of STDP to arbitrary time courses of pre- and postsynaptic activities, i.e. its nonlinear filter properties. We demonstrate this for the simple example of small periodic modulations of pre- and postsynaptic firing rates for which our model can be solved. It predicts synaptic strengthening for synchronous rate modulations. Modifications are dominant in the theta frequency range, a result which underlines the well known relevance of theta activities in hippocampus and cortex for learning. We also find emphasis of specific baseline spike rates and suppression for high background rates. The latter suggests a mechanism of network activity regulation inherent in STDP. Furthermore, our novel formulation provides a general framework for investigating the joint dynamics of neuronal activity and the CD of STDP in both spike-based as well as rate-based neuronal network models.

## 1 Introduction

During the past decade the effects of exact spike timing on the change of synaptic connectivity have been studied extensively. *In vitro* studies have shown that the induction of long-term potentiation (LTP) requires the presynaptic input to a cell to precede the postsynaptic output and vice versa for long-term depression (LTD) (see [1, 2, 3]). This phenomenon has been termed spike timing-dependent plasticity (STDP) and emphasizes the importance of a causal order in neuronal signaling. Thereby it extends pure Hebbian learning, which requires only the coincidence of pre- and postsynaptic activity. Consequently, experiments have shown an asymmetric exponential dependence on the timing of spike pairs and a molecular mechanism mostly dependent on the influx of $Ca^{2+}$ (see [4, 5] for reviews). Further, when induced with more complex spike trains, synaptic modification shows nonlinearities ([6, 7, 8]) indicating the influence of short-term plasticity.

Theoretical approaches to STDP cover studies using the asymmetric pair-based STDP window as a lookup table, more biophysical models based on synaptic and neuronal variables, and sophisticated kinetic models (for a review see [9]). Recently, the experimentally observed influence of the postsynaptic membrane potential (e.g. [10]) has also been taken into account ([11]).

Our approach is based on differential Hebbian learning ([12, 13]), which generates asymmetric timing windows similar to STDP ([14]) depending on the shape of the back-propagating action

---

[*] Postal correspondence should be addressed to Universität Bremen, Fachbereich 1, Institut für Theoretische Physik, Abt. Neurophysik, Postfach 330 440, D-28334 Bremen, Germany

potential ([15]). We extend it with a mechanism for activating learning by an increase in postsynaptic activity, because both the induction of LTP and LTD require $[Ca^{2+}]$ to exceed a threshold ([16]). Moreover, we include a mechanism for adaptive suppression on both synaptic sides, similar to the model in [7]. Finally, we for simplicity assume that both the presynaptic and the postsynaptic side function as low-pass filters; a spike leaves a fast increasing and exponentially decaying trace. Together, we propose a set of differential equations, which captures the contribution dynamics (CD) of pre- and postsynaptic activities to STDP, thereby describing synaptic plasticity as a filter.

Our framework reproduces experimental findings from two recent in vitro studies in the visual cortex and the hippocampus in most details. Furthermore, it proves to be particularly suitable for the analysis of the susceptibility of STDP to pre- and postsynaptic rate modulations. This is demonstrated by an analysis of synaptic changes depending on oscillatory modulations of baseline firing rates.

## 2 Formulation of the model

We use a variant of the classical differential Hebbian learning assuming a change of synaptic connectivity $w$, which is dependent on the presynaptic activity trace $y_{\mathrm{pre}}$ and the temporal derivative of the postsynaptic activity trace $y_{\mathrm{post}}$:

$$\dot{w}(t) = c_w \, y_{\mathrm{pre}}(t)\dot{y}_{\mathrm{post}}(t) \, . \tag{1}$$

$c_w$ denotes a constant learning rate. An illustration of this learning rule for pairs of spikes is given in Figure 1B. For simplicity, we assume these activity traces to be abstract low-pass filtered versions of neuronal activity $x$ in the presynaptic and postsynaptic cells, e.g. the concentration of $Ca^{2+}$ or the amount of bound glutamate:

$$\dot{y}_{\mathrm{pre}}(t) = u_{\mathrm{pre}}(t) \cdot x_{\mathrm{pre}}(t) - \frac{y_{\mathrm{pre}}(t)}{\tau_{\mathrm{pre}}} \tag{2}$$

$$\dot{y}_{\mathrm{post}}(t) = u_{\mathrm{post}}(t)z(t) \cdot x_{\mathrm{post}}(t) - \frac{y_{\mathrm{post}}(t)}{\tau_{\mathrm{post}}} \, . \tag{3}$$

The dynamics of the $y$'s are characterized by their respective time constants $\tau_{\mathrm{pre}}$ and $\tau_{\mathrm{post}}$. The contribution of each spike is regulated by a suppressing *attenuation factor* $u$ pre- and postsynaptically. On the postsynaptical side an additional *activation factor* $z$ "enables" the synapse to learn. The dynamics of $u$ and $z$ are discussed below. $x$ represents neuronal activity which can be either a time-continuous firing rate or spike trains given by series of $\delta$ pulses

$$x_{\mathrm{pre, post}}(t) = \sum_i \delta(t - t^i_{\mathrm{pre, post}}) \quad , \tag{4}$$

which allows analytical investigations of the properties of our model. Note that formally $x(t)$ has then to be taken as $x(t+0)$. An illustrating overview over the different parts of the model with sample trajectories is shown in Figure 1A.

We define the relative change of synaptic connectivity after after a period $T$ from Equation (1) as

$$\Delta w = \frac{w(t_0 + T)}{w(t_0)} - 1 = \frac{c_w}{w(t_0)} \int_T y_{\mathrm{pre}}\dot{y}_{\mathrm{post}} \, dt \, . \tag{5}$$

The dependence on the initial synaptic strength $w(t_0)$ as observed in [3, 8] shall not be discussed here, but can easily be achieved by making the learning rate $c_w$ in Equation (1) $w$-dependent. Here, $w(t_0)$ is chosen to be 1.

Ignoring attenuation and activation, a single pair of spikes at temporal distance $\Delta t$ analytically yields the typical STDP window (see Figure 2A and 3A):

$$\Delta w(\Delta t) = \begin{cases} c_w \left(1 - \frac{\tau_{\mathrm{pre}}}{\tau_{\mathrm{pre}} + \tau_{\mathrm{post}}}\right) e^{-\Delta t/\tau_{\mathrm{pre}}} & \text{for} \quad \Delta t > 0 \\ c_w \cdot \frac{\tau_{\mathrm{pre}}}{\tau_{\mathrm{pre}} + \tau_{\mathrm{post}}} e^{-\Delta t/\tau_{\mathrm{post}}} & \text{for} \quad \Delta t < 0 \end{cases} \tag{6}$$

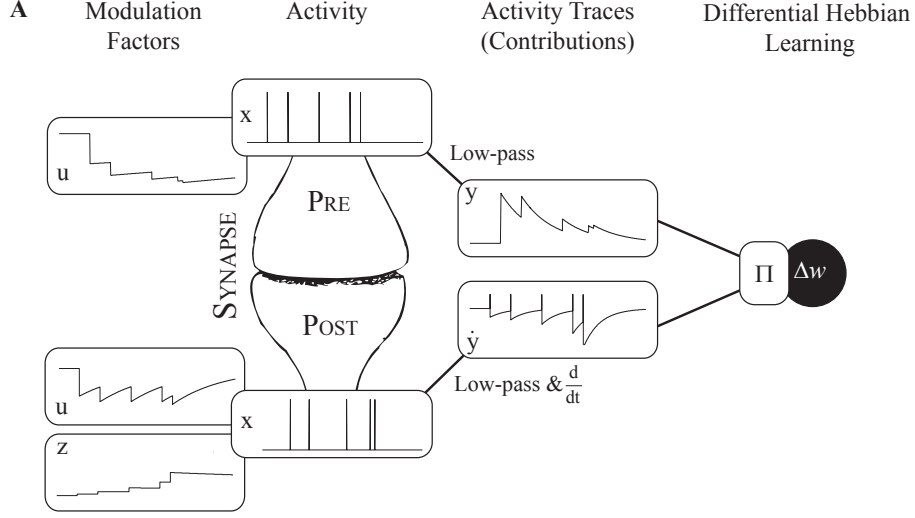

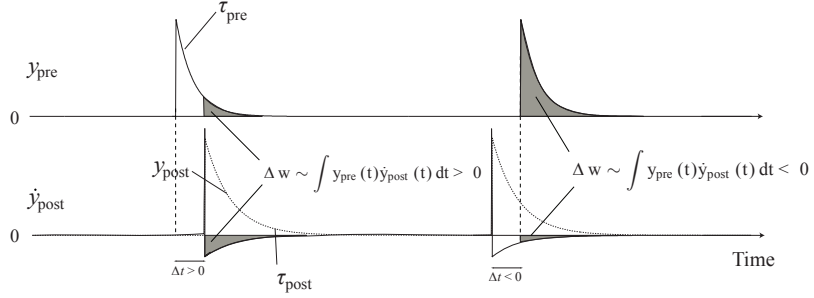

Figure 1: Schematic illustration of differential Hebbian learning with contribution dynamics. **A**: Pre- and postsynaptic activity ($x$, second column) is modulated (attenuated with $u$, activated with $z$, first column) and filtered ($y$, third column) before it contributes to differential Hebbian learning ($w$, fourth column). **B**: Spike pair example for differential Hebbian learning. Left: a presynaptic spike trace ($y_{\text{pre}}$) preceding a postsynaptic spike trace ($y_{\text{post}}$, dotted line) yields a synaptic strengthening due to the initially positive postsynaptic contribution ($\dot{y}_{\text{post}}$, solid line), which is always stronger than the following negative part. Right: for the reverse timing the positive presynaptic contribution is only multiplied with the negative postsynaptic trace (right). Areas contributing to learning are shaded.

The importance of adaptive suppressing mechanisms for synaptic plasticity has experimentally been shown by Froemke and colleagues ([7, 6]). Therefore, we down-regulate the contribution of the spikes to the activity traces $y$ in Equation (2) and (3) with an attenuation factor $u$ on both pre- and postsynaptic sides:

$$\dot{u}_{\text{pre}} = \frac{1}{\tau_{\text{pre}}^{\text{rec}}}(1 - u_{\text{pre}}) - c_{\text{pre}} u_{\text{pre}} x_{\text{pre}} \tag{7}$$

$$\dot{u}_{\text{post}} = \frac{1}{\tau_{\text{post}}^{\text{rec}}}(1 - u_{\text{post}}) - c_{\text{post}}(u_{\text{post}} - u_0)x_{\text{post}} \quad . \tag{8}$$

This should be understood as an abstract representation of for instance the depletion of transmitters in the presynaptic bouton ([17]) or the frequency-dependent spike attenuation in dendritic spines ([18]), respectively. These recover with their time constants $\tau^{\text{rec}}$ and are bound between $u_0$ and 1.

For the presynaptic side we assume in the following $u_0^{\text{pre}} = 0$, so we abbreviate $u_0 = u_0^{\text{post}}$. The constants $c_{\text{pre, post}} \in [0, 1]$ denote the impact a spike has on the relaxed synapse.

In several experiments it has been shown that a single spike is not sufficient to induce synaptic modification ([10, 8]). Therefore, we introduce a spike-induced postsynaptic activation factor $z$

$$\dot{z} = c_{\text{act}} x_{\text{post}} z - \alpha (z - z_0)^2 , \tag{9}$$

which enhances the contribution of a postsynaptic spike to the postsynaptic trace, e.g. by the removal of the $Mg^{2+}$ block from postsynaptic NMDA receptors ([19, 5]). The nonlinear positive feedback is introduced to describe strong enhancing effects as for instance autocatalytic mechanisms, which have been suggested to play a role in learning on several time-scales ([20, 21]). The activation $z$ decays hyperbolically to a lower bound $z_0$ and the contribution of a spike is weighted with the constant $c_{\text{act}}$.

## 3  Comparison to experiments

In order to evaluate our model we implemented experimental stimulation protocols from *in vitro* studies on synapses of the visual cortex ([7]) and the hippocampus ([8]) of rats. In both studies, simple pairs of spikes and more complex spike trains were artificially elicited in the presynaptic and the postsynaptic cell and the induced change of synaptic connectivity was recorded.

Froemke and colleagues ([7]) focused on the effects of spike bursts on synaptic modification in the visual cortex. In addition to the classical STDP pairing protocol – a presynaptic spike preceding or following a postsynaptic spike after a specific time $\Delta t$ – four other experimental protocols (see Figure 2B to E) were performed: (1) 5-5 bursts with five spikes of a certain frequency on both synaptic sides, where the postsynaptic side follows the presynaptic side, (2) presynaptic 100 Hz bursts with $n$ spikes following one postsynaptic spike (post-$n$-pre), (3) presynaptic 100 Hz bursts with different numbers of spikes followed by one postsynaptic spike ($n$-pre-post) and (4) a post-pre pair with varying number of following postsynaptic spikes (post-pre-$n$-post).

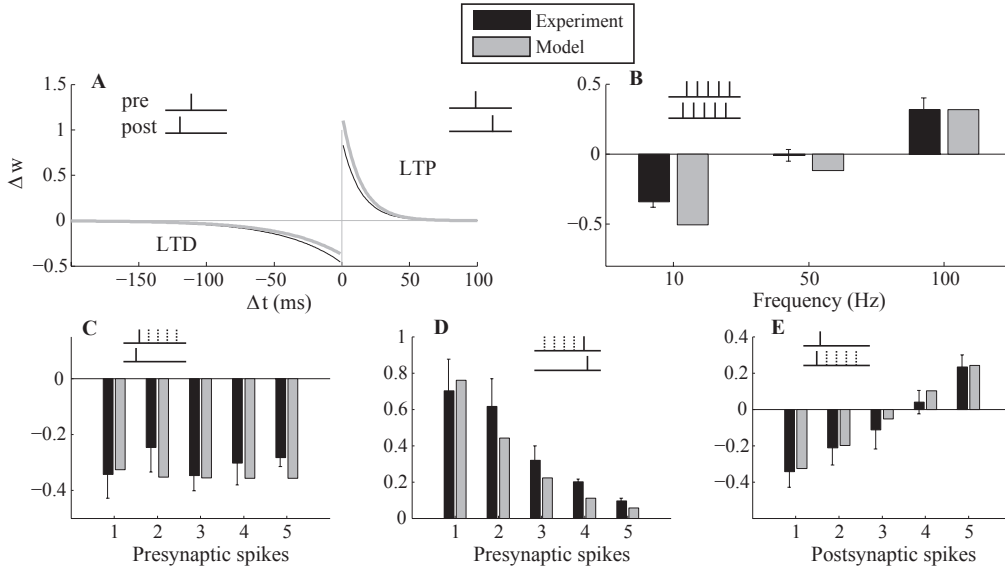

Figure 2: Differential Hebbian learning with CD reproduces synaptic modification induced with STDP spike patterns in visual cortex. Data taken from [7], personal communication. **A**: experimental fit and model prediction with Equation (6) of pair-based STDP. **B**: dependence of synaptic modifications on the frequency of 5-5 bursts with presynaptic spikes following postsynaptic spikes by 6 ms. **C, D** and **E**: synaptic modification induced by post-$n$-pre, $n$-pre-post and post-pre-$n$-post 100 Hz spike trains.

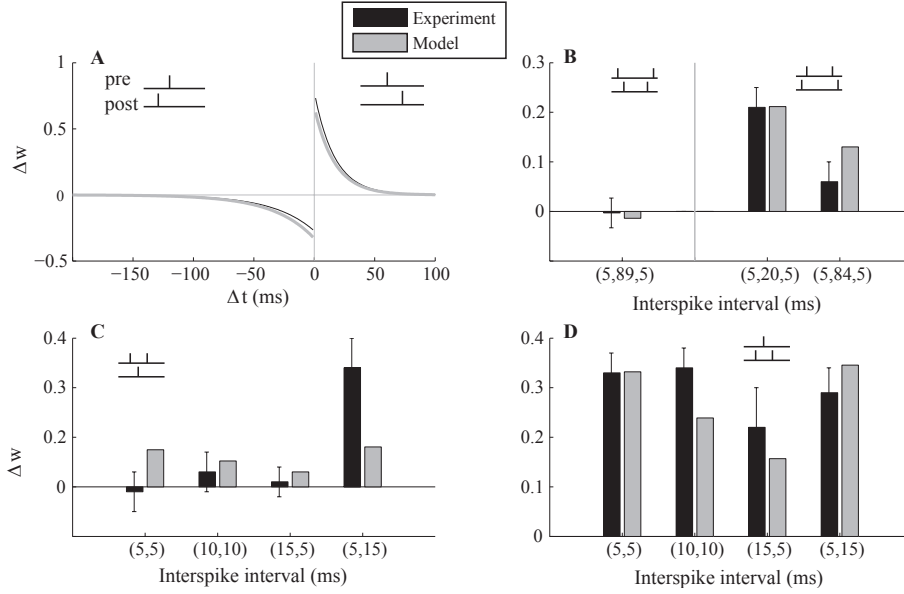

Figure 3: Differential Hebbian learning with CD reproduces synaptic modification induced with STDP spike patterns in hippocampus. Data taken from [8] as reported in [22]. **A**: experimental fit and model prediction with Equation (6) of pair-based STDP. **B**: quadruplet protocol. **C** and **D**: post-pre-post and pre-post-pre triplet protocol for different interspike intervals.

Table 1: Parameters and evaluation results for the data sets from visual cortex ([7]) and hippocampus ([8]). $E$: normalized mean-square error, $S$: ratio of correctly predicted signs of synaptic modification.

|  | $c_{\text{pre}}$ | $c_{\text{post}}$ | $c_{\text{act}}$ | $\tau_{\text{pre}}^{\text{rec}}$ [s] | $\tau_{\text{post}}^{\text{rec}}$ [s] | $\alpha$ | $u_0$ | $z_0$ | $E$ | $S$ |
|---|---|---|---|---|---|---|---|---|---|---|
| Visual cortex | 0.9 | 1 | 1.5 | 2 | 0.2 | 1 | 0.01 | 1 | 4.04 | 18/18 |
| Hippocampus | 0.6 | 0.4 | 3.5 | 0.5 | 0.5 | 1 | 0.7 | 0.2 | 2.16 | 10/11 |

In the hippocampal study of Wang et al. ([8]) synaptic modification induced by triplets (pre-post-pre and post-pre-post) and quadruplets (pre-post-post-pre and post-pre-pre-post) of spikes was measured while the respective interspike intervals were varied. (see Figure 3B to D).

As a first step we took the time constants from the experimentally measured pair-based STDP windows as our low-pass filter time constants (see Equation 6). They remained constant for each data set: (1) $\tau_{\text{pre}} = 13.5\,\text{ms}$ and $\tau_{\text{post}} = 42.8\,\text{ms}$ for [7], (2) $\tau_{\text{pre}} = 16.8\,\text{ms}$ and $\tau_{\text{post}} = 33.7\,\text{ms}$ for [8] (taken from [23] since not present in the study). Next, we chose the learning rate $c_w$ in Equation (6) to fit the synaptic change for the pairing protocol: (1) $c_w = 1.56$ for the visual cortex data, (2) $c_w = 0.99$ for the hippocampal data set. The remaining parameters were estimated manually within biologically plausible ranges and are shown in Table 1. The model was then applied to the more complex stimulation protocols by solving the differential equations semi-analytically, i.e. separately for every spike and the following interspike interval. As measure for the prediction error of our model we used the normalized mean-square error $E$

$$E = \frac{1}{N} \sum_{i=1}^{N} \left( \frac{\Delta w_i^{\text{exp}} - \Delta w_i^{\text{mod}}}{\sigma_i} \right)^2 , \tag{10}$$

where $\Delta w_i^{\text{exp}}$ and $\Delta w_i^{\text{mod}}$ are the experimentally measured and the predicted modifications of synaptic strength in the $i$th experiment; $N$ is the number of data points ($N = 18$ for the visual cortex data set, $N = 11$ for the hippocampal data set). $\sigma_i$ is the standard error of the mean of the experimental data. Additionally we counted the number of correctly predicted signs $S$ of synaptic modification, i.e. induced depression or potentiation. The prediction error for both data sets is shown in Table 1.

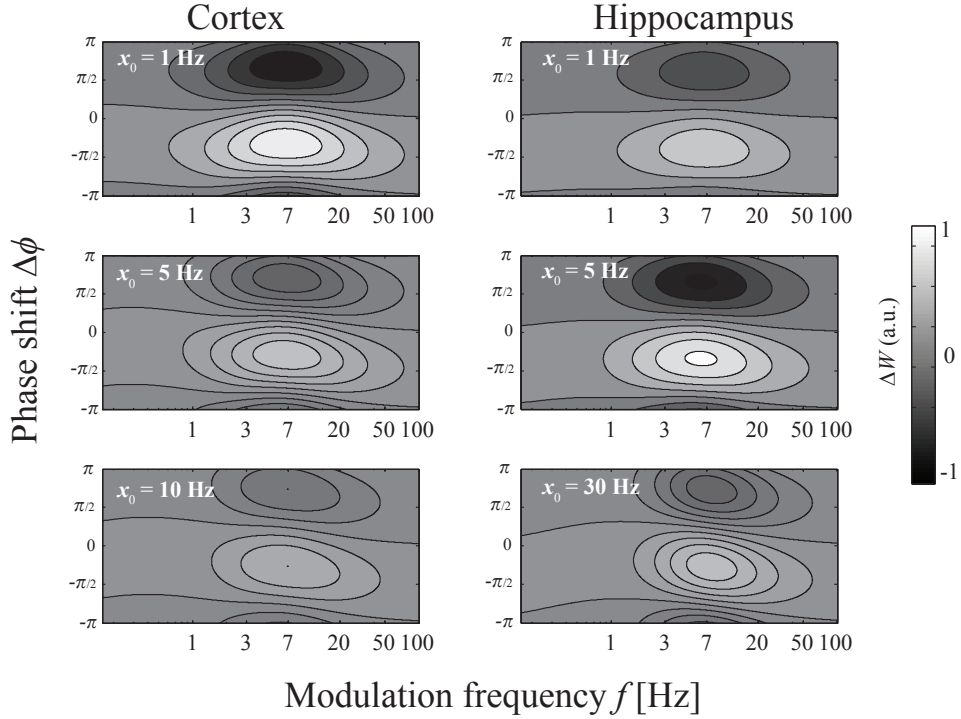

Figure 4: Synaptic change depending on frequency $f$ and phase shift $\Delta\phi$ of pre- and postsynaptic rate modulations for different baseline rates $x_0$. The color codes are identical within each column and in arbitrary units. Note the strong suppression with increasing baseline rate for cortical synapses which is due to strong attenuation effects of pre- and postsynaptic contributions. It is weaker for hippocampal synapses because we found the postsynaptic attenuation to be bounded ($u_0 = 0.7$).

## 4 Phase, frequency and baseline rate dependence of STDP with contribution dynamics

As shown in the previous section our model can reproduce the experimental findings of synaptic weight changes in response to spike sequences surprisingly well and yields better fits than former studies (e.g. [22]). The proposed framework, however, is not restricted to spike sequences but allows to investigate synaptic changes depending on arbitrary pre- and postsynaptic activities. For instance it could be used for investigations of the plasticity effects in simulations with inhomogeneous Poisson processes. Taking $x(t)$ to be firing rates of Poissonian spike trains our account of STDP represents a useful approximation for the expected changes of synaptic strength depending on the time courses of $x_{\mathrm{pre}}$ and $x_{\mathrm{post}}$ (compare e.g. [24]). Therefore our model can serve also as building block in rate based network models for investigation of the joint dynamics of neuronal activities and synaptic weights.

Here, we demonstrate the benefit of our approach for determining the filter properties of STDP subject to CD, i.e. we use the equations together with the parameters from the experiments for determining the dependency of weight changes on frequency, relative phase $\Delta\phi$ and baseline rates of modulated pre- and postsynaptic firing rates. While for substantial modulations of firing rates the nonlinearities are difficult to be treated analytically, for small periodical modulations around a baseline rate $x_0$ the corresponding synaptic changes can be calculated analytically. This is done by considering

$$x_{\mathrm{pre}}(t) = x_0 + \varepsilon \cos(2\pi f t) \quad \text{and} \quad x_{\mathrm{post}}(t) = x_0 + \varepsilon \cos(2\pi f t - \Delta\phi), \qquad (11)$$

which for small $\varepsilon < x_0$ allows linearization of all equations from which one obtains $\Delta W = \Delta w/(T\varepsilon_{\mathrm{pre}}\varepsilon_{\mathrm{post}})$, where $T = 1/f = 2\pi/\omega$ is the period of the respective oscillations. Neglect-

ing transients this finally yields the expected weight changes per unit time. Though lengthy the calculations are straightforward and presented in the supplementary material. We here show only the exact result for the case of constant $u = 1$ and $z = 1$:

$$\Delta W = \frac{\omega \tau_{\text{pre}} \tau_{\text{post}} \sqrt{\omega^2 (\tau_{\text{post}} - \tau_{\text{pre}})^2 + (1 + \omega^2 \tau_{\text{pre}} \tau_{\text{post}})^2}}{2(1 + \tau_{\text{pre}}^2 \omega^2)(1 + \tau_{\text{post}}^2 \omega^2)} \cdot \sin\left(\Delta \phi + \arctan \frac{\omega(\tau_{\text{post}} - \tau_{\text{pre}})}{1 + \omega^2 \tau_{\text{pre}} \tau_{\text{post}}}\right) \quad (12)$$

The analytical results for the case with CD are shown graphically in Figure 4 using the parameters from cortex and hippocampus, respectively (see Tab. 1). These plots contain the main findings: (1) rate modulations in the theta frequency range ($\simeq$ 7Hz) lead to strongest synaptic changes, (2) also for phase-zero synchronous rate modulations weight changes are positive, (3) in hippocampus maximal weight change magnitudes occur at baseline rates around 5 Hz, and (4) for high baseline rates weight changes become suppressed ($\sim 1/x_0$ for the hippocampus, $\sim 1/x_0^2$ for the visual cortex). Numerical simulations with finite rate modulations were found to confirm these analytical predictions surprisingly well. Also for the nonlinear regime and Poissionian spike trains deviations remained moderate.

## 5  Discussion

STDP has been proposed to represent a fundamental mechanism underlying learning and many models explored its computational role (examples are [25, 26, 27]). In contrast, research targeting the computational roles of dynamical phenomena inherent in STDP are in the beginning (see [9]). Here, we here formulated a minimal, yet biologically plausible model including the dynamics of how neuronal activity contributes to STDP. We found that our model reproduces the synaptic changes in response to spike sequences in experiments in cortex and hippocampus with high accuracy.

Using the corresponding parameters our model predicts weight changes depending on temporal structures in the pre- and postsynaptic activities including spike sequences and varying firing rates. When applied to pre- and postsynaptic rate modulations our approach quantifies synaptic changes depending on frequency and phase shifts between pre- and postsynaptic activities. A rigorous perturbation analysis of our model reveals that the dynamical filter properties of STDP make weight changes sensitively dependent on combinations of specific features of pre- and postsynaptic signals.

In particular, our analysis indicates that both cortical as well as hippocampal STDP is most susceptible for modulations in the theta frequency range. It predicts the dependency of synaptic changes on pre- and postsynaptic phase relations of rate modulations. These results are in line with experimental results on the relation of theta rhythms and learning. For instance in hippocampus it is well established that theta oscillations are relevant for learning (for a recent paper see [28]). Furthermore, spike activities in hippocampus exhibit specific phase relations with the theta rhythm (for a review see [29]). Also, it has been found that during learning cortex and hippocampus tend to synchronize with particular phase relations that depend on the novelty of the item to be learned ([30]). The results presented here underline these findings and make testable predictions for the corresponding synaptic changes.

Also, we find potentiation for zero phase differences and strong attenuation of weight changes at large baseline rates which is particularly strong for cortical synapses. This finding suggests a mechanism for restricting weight changes with high activity levels and that STDP is de facto switched off when large firing rates are required for the execution of a function as opposed to learning phases; during the latter baseline rates should be rather low, which is particularly relevant in cortex. While for cortical synapses our analysis predicts that very low baseline activities are contributing most to weight changes, in hippocampus synaptic modifications peak at baseline firing rates $x_0$ around 5 Hz, which suggests that $x_0$ can control learning.

Our study suggests that the filter properties of STDP originating from the dynamics of pre- and postsynaptic activity contributions are in fact exploited for learning in the brain. In particular, shifts in baseline rates, as well as the frequency and the respective phases of pre- and postsynaptic rate modulations induced by theta oscillations could be tuned to match the values that make STDP most susceptible for synaptic modifications. A fascinating possibility thereby is that these features could be used to control the learning rate which would represent a novel mechanism in addition to other control signals as e.g. neuromodulators.

# References

[1] W. Levy and O. Steward. Temporal contiguity requirements for long-term associative potentiation/depression in the hippocampus. *Neuroscience*, 8(4):791–797, 1983.

[2] H. Markram, J. Lubke, M. Frotscher, and B. Sakmann. Regulation of synaptic efficacy by coincidence of postsynaptic APs and EPSPs. *Science*, 1997.

[3] G. Q. Bi and M. M. Poo. Synaptic modifications in cultured hippocampal neurons: dependence on spike timing, synaptic strength, and postsynaptic cell type. *Journal of Neuroscience*, 18(24):10464–72, 1998.

[4] P. J. Sjöström, E. A. Rancz, A. Roth, and M. Häusser. Dendritic excitability and synaptic plasticity. *Physiological Reviews*, 88(2):769–840, 2008.

[5] N. Caporale and Y. Dan. Spike timing–dependent plasticity: a Hebbian learning rule. *Annual Review in Neuroscience*, 2008.

[6] R. C. Froemke and Y. Dan. Spike-timing-dependent synaptic modification induced by natural spike trains. *Nature*, 2002.

[7] R. C. Froemke, I. A. Tsay, M. Raad, J. D. Long, and Y. Dan. Contribution of individual spikes in burst-induced long-term synaptic modification. *Journal of Neurophysiology*, 95(3):1620–9, 2006.

[8] H. X. Wang, R. C. Gerkin, D. W. Nauen, and G. Q. Bi. Coactivation and timing-dependent integration of synaptic potentiation and depression. *Nature Neuroscience*, 8(2):187–93, 2005.

[9] A. Morrison, M. Diesmann, and W. Gerstner. Phenomenological models of synaptic plasticity based on spike timing. *Biological Cybernetics*, 98(6):459–78, 2008.

[10] P. J. Sjöström, G. G. Turrigiano, and S. B. Nelson. Rate, timing, and cooperativity jointly determine cortical synaptic plasticity. *Neuron*, 32(6):1149–1164, 2001.

[11] C. Clopath, L. Büsing, E. Vasilaki, and W. Gerstner. Connectivity reflects coding: a model of voltage-based STDP with homeostasis. *Nature Neuroscience*, 13(3):344–52, 2010.

[12] B. Kosco. Differential Hebbian learning. *AIP Conference Proceedings 151 on Neural Networks for Computing*, 1987.

[13] A. H. Klopf. A drive-reinforcement model of single neuron function: An alternative to the Hebbian neuronal model. *AIP Conference Proceedings*, 151(1):265–270, 1986.

[14] P. D. Roberts. Computational consequences of temporally asymmetric learning rules: I. differential Hebbian learning. *Journal of Computational Neuroscience*, 7(3):235–246, 1999.

[15] A. Saudargiene, B. Porr, and F. Wörgötter. How the shape of pre-and postsynaptic signals can influence STDP: a biophysical model. *Neural Computation*, 2004.

[16] T. Nevian and B. Sakmann. Spine Ca2+ signaling in spike-timing-dependent plasticity. *Journal of Neuroscience*, 26(43):11001–13, 2006.

[17] M. V. Tsodyks and H. Markram. The neural code between neocortical pyramidal neurons depends on neurotransmitter release probability. *Proceedings of the National Academy of Sciences*, 94(2):719–723, 1997.

[18] E. Tanaka, H. Higashi, and S. Nishi. Membrane properties of guinea pig cingulate cortical neurons in vitro. *J Neurophysiol*, 65(4):808–821, 1991.

[19] L. Nowak, P. Bregestovski, P. Ascher, A. Herbet, and A. Prochiantz. Magnesium gates glutamate-activated channels in mouse central neurones. *Nature*, 307(5950):462–5, 1984.

[20] J. E. Lisman. A Mechanism for Memory Storage Insensitive to Molecular Turnover: A Bistable Autophosphorylating Kinase. *Proceedings of the National Academy of Sciences*, 82(9):3055–3057, 1985.

[21] U. S. Bhalla and R. Iyengar. Emergent Properties of Networks of Biological Signaling Pathways. *Science*, 283(5400):381–387, 1999.

[22] J. P. Pfister and W. Gerstner. Triplets of spikes in a model of spike timing-dependent plasticity. *Journal of Neuroscience*, 26(38):9673–82, 2006.

[23] G. Bi and M. Poo. Synaptic modification by correlated activity: Hebb's postulate revisited. *Annual Review of Neuroscience*, 24:139–66, 2001.

[24] M. Tsodyks, K. Pawelzik, and H. Markram. Neural networks with dynamic synapses. *Neural Computation*, 10(4):821–35, 1998.

[25] M. Lengyel, J. Kwag, O. Paulsen, and P. Dayan. Matching storage and recall: hippocampal spike timing-dependent plasticity and phase response curves. *Nature Neuroscience*, 8(12):1677–83, 2005.

[26] F. Wörgötter and B. Porr. Temporal sequence learning, prediction, and control: a review of different models and their relation to biological mechanisms. *Neural Computation*, 17(2):245–319, 2005.

[27] E. M. Izhikevich. Solving the distal reward problem through linkage of STDP and dopamine signaling. *Cerebral Cortex*, 17(10):2443–52, 2007.

[28] U. Rutishauser, I. B. Ross, A. N. Mamelak, and E. M. Schuman. Human memory strength is predicted by theta-frequency phase-locking of single neurons. *Nature*, 464(7290):903–7, 2010.

[29] Y. Yamaguchi, N. Sato, H. Wagatsuma, Z. Wu, C. Molter, and Y. Aota. A unified view of theta-phase coding in the entorhinal-hippocampal system. *Current Opinion in Neurobiology*, 17(2):197–204, 2007.

[30] A. Jeewajee, C. Lever, S. Burton, J. O'Keefe, and N. Burgess. Environmental novelty is signaled by reduction of the hippocampal theta frequency. *Hippocampus*, 18(4):340–8, 2008.

